# Are Hopfield Networks Faster Than Conventional Computers?

**Ian Parberry**[*] and **Hung-Li Tseng**[†]
Department of Computer Sciences
University of North Texas
P.O. Box 13886
Denton, TX 76203–6886

## Abstract

It is shown that conventional computers can be exponentially faster
than planar Hopfield networks: although there are planar Hopfield
networks that take exponential time to converge, a stable state of an
arbitrary planar Hopfield network can be found by a conventional
computer in polynomial time. The theory of $\mathcal{PLS}$-completeness
gives strong evidence that such a separation is unlikely for nonpla-
nar Hopfield networks, and it is demonstrated that this is also the
case for several restricted classes of nonplanar Hopfield networks,
including those who interconnection graphs are the class of bipar-
tite graphs, graphs of degree 3, the dual of the knight's graph, the
8-neighbor mesh, the hypercube, the butterfly, the cube-connected
cycles, and the shuffle-exchange graph.

## 1 Introduction

Are Hopfield networks faster than conventional computers? This apparently
straightforward question is complicated by the fact that conventional computers
are universal computational devices, that is, they are capable of simulating any
discrete computational device including Hopfield networks. Thus, a conventional
computer could in a sense cheat by imitating the fastest Hopfield network possible.

---

[*]Email: ian@cs.unt.edu. URL: http://hercule.csci.unt.edu/ian.
[†]Email: htseng@ponder.csci.unt.edu.

But the question remains, is it faster for a computer to imitate a Hopfield network, or to use other computational methods? Although the answer is likely to be different for different benchmark problems, and even for different computer architectures, we can make our results meaningful in the long term by measuring *scalability*, that is, how the running time of Hopfield networks and conventional computers increases with the size of any benchmark problem to be solved.

Stated more technically, we are interested in the computational complexity of the *stable state problem* for Hopfield networks, which is defined succinctly as follows: given a Hopfield network, determine a stable configuration. As previously stated, this stable configuration can be determined by imitation, or by other means. The following results are known about the scalability of Hopfield network imitation. Any imitative algorithm for the stable state problem must take exponential time on *some* Hopfield networks, since there exist Hopfield networks that require exponential time to converge (Haken and Luby [4], Goles and Martinez [2]). It is unlikely that even non-imitative algorithms can solve the stable state problem in polynomial time, since the latter is $\mathcal{PLS}$-complete (Papadimitriou, Schäffer, and Yannakakis [9]). However, the stable state problem is more difficult for some classes of Hopfield networks than others. Hopfield networks will converge in polynomial time if their weights are bounded in magnitude by a polynomial of the number of nodes (for an expository proof see Parberry [11, Corollary 8.3.4]). In contrast, the stable state problem for Hopfield networks whose interconnection graph is bipartite is $\mathcal{PLS}$-complete (this can be proved easily by adapting techniques from Bruck and Goodman [1]) which is strong evidence that it too requires superpolynomial time to solve even with a nonimitative algorithm.

We show in this paper that although there exist planar Hopfield networks that take exponential time to converge in the worst case, the stable state problem for planar Hopfield networks can be solved in polynomial time by a non-imitative algorithm. This demonstrates that imitating planar Hopfield networks is exponentially slower than using non-imitative algorithmic techniques. In contrast, we discover that the stable state problem remains $\mathcal{PLS}$-complete for many simple classes of nonplanar Hopfield network, including bipartite networks, networks of degree 3, and some networks that are popular in neurocomputing and parallel computing.

The main part of this manuscript is divided into four sections. Section 2 contains some background definitions and references. Section 3 contains our results about planar Hopfield networks. Section 4 describes our $\mathcal{PLS}$-completeness results, based on a pivotal lemma about a nonstandard type of graph embedding.

## 2   Background

This section contains some background which are included for completeness but may be skipped on a first reading. It is divided into two subsections, the first on Hopfield networks, and the second on $\mathcal{PLS}$-completeness.

### 2.1   Hopfield Networks

A *Hopfield network* [6] is a discrete neural network model with symmetric connections. Each processor in the network computes a hard binary weighted threshold

function. Only one processor is permitted to change state at any given time. That processor becomes active if its excitation level exceeds its threshold, and inactive otherwise. A Hopfield network is said to be in a *stable state* if the states of all of its processors are consistent with their respective excitation levels. It is well-known that all Hopfield networks converge to a stable state. The proof defines a measure called *energy*, and demonstrates that energy is positive but decreases with every computation step. Essentially then, a Hopfield network finds a local minimum in some energy landscape.

## 2.2 $\mathcal{PLS}$-completeness

While the theory of $\mathcal{NP}$-completeness measures the complexity of global optimization, the theory of $\mathcal{PLS}$-completeness developed by Johnson, Papadimitriou, and Yannakakis [7] measures the complexity of local optimization. It is similar to the theory of $\mathcal{NP}$-completeness in that it identifies a set of difficult problems known collectively as $\mathcal{PLS}$-*complete* problems. These are difficult in the sense that if a fast algorithm can be developed for any $\mathcal{PLS}$-complete problem, then it can be used to give fast algorithms for a substantial number of other local optimization problems including many important problems for which no fast algorithms are currently known. Recently, Papadimitriou, Schäffer, and Yannakakis [9] proved that the problem of finding stable states in Hopfield networks is $\mathcal{PLS}$-complete.

## 3 Planar Hopfield Networks

A planar Hopfield network is one whose interconnection graph is planar, that is, can be drawn on the Euclidean plane without crossing edges. Haken and Luby [4] describe a planar Hopfield network that provably takes exponential time to converge, and hence any imitative algorithm for the stable state problem must take exponential time on *some* Hopfield network. Yet there exists a nonimitative algorithm for the stable state problem that runs in polynomial time on *all* Hopfield networks:

**Theorem 3.1** *The stable state problem for Hopfield networks with planar interconnection pattern can be solved in polynomial time.*

PROOF: (Sketch.) The proof follows from the fact that the maximal cut in a planar graph can be found in polynomial time (see, for example, Hadlock [3]), combined with results of Papadimitriou, Schäffer, and Yannakakis [9]. □

## 4 $\mathcal{PLS}$-completeness Results

Our $\mathcal{PLS}$-completeness results are a straightforward consequence of a new result that characterizes the difficulty of the stable state problem of an arbitrary class of Hopfield networks based on a graph-theoretic property of their interconnection patterns. Let $G = (V, E)$ and $H = (V', E')$ be graphs. An *embedding* of $G$ into $H$ is a function $f: V \rightarrow 2^{V'}$ such that the following properties hold. (1) For all $v \in V$, the subgraph of $H$ induced by $f(v)$ is connected. (2) For all $(u, v) \in E$, there exists a path (which we will denote $f(u, v)$) in $H$ from a member of $f(u)$ to a member of $f(v)$. (3) Each vertex $w \in H$ is used at most once, either as a member of $f(v)$

for some $v \in V$, or as an internal vertex in a path $f(u,v)$ for some $u,v \in V$. The graph $G$ is called the *guest* graph, and $H$ is called the *host* graph. Our definition of embedding is different from the standard notion of embedding (see, for example, Hong, Mehlhorn, and Rosenberg [5]) in that we allow the image of a single guest vertex to be a *set* of host vertices, and we insist in properties (2) and (3) that the images of guest edges be distinct paths. The latter property is crucial to our results, and forms the major difficulty in the proofs.

Let $S, T$ be sets of graphs. $S$ is said to be *polynomial-time embeddable* into $T$, written $S \leq_e T$, if there exists polynomials $p_1(n), p_2(n)$ and a function $f$ with the following properties: (1) $f$ can be computed in time $p_1(n)$, and (2) for every $G \in S$ with $n$ vertices, there exists $H \in T$ with at most $p_2(n)$ vertices such that $G$ can be embedded into $H$ by $f$. A set $S$ of graphs is said to be *pliable* if the set of all graphs is polynomial-time embeddable into $S$.

**Lemma 4.1** *If $S$ is pliable, then the problem of finding a stable state in Hopfield networks with interconnection graphs in $S$ is $\mathcal{PLS}$-complete.*

PROOF: (Sketch.) Let $S$ be a set of graphs with the property that the set of all graphs is polynomial-time embeddable into $S$. By the results of Papadimitriou, Schäffer, and Yannakakis [9], it is enough to show that the max-cut problem for graphs in $S$ is $\mathcal{PLS}$-complete.

Let $G$ be an arbitrary labeled graph. Suppose $G$ is embedded into $H \in S$ under the polynomial-time embedding. For each edge $e$ in $G$ of cost $c$, select one edge from the path connecting the vertices in $f(e)$ and assign it cost $c$. We call this special edge $f'(e)$. Assign all other edges in the path cost $-\infty$. For all $v \in V$, assign the edges linking the vertices in $f(v)$ a cost of $-\infty$. Assign all other edges of $H$ a cost of zero.

It can be shown that every cut in $G$ induces a cut of the same cost in $H$, as follows. Suppose $C \subseteq E$ is a cut in $G$, that is, a set of edges that if removed from $G$, disconnects it into two components containing vertices $V_1$ and $V_2$ respectively. Then, removing vertices $f'(C)$ and all zero-cost edges from $H$ will disconnect it into two components containing vertices $f(V_1)$ and $f(V_2)$ respectively. Furthermore, each cut of positive cost in $H$ induces a cut of the same cost in $G$, since a positive cost cut in $H$ cannot contain any edges of cost $-\infty$, and hence must consist only of $f'(e)$ for some edges $e \in E$. Therefore, every max-cost cut in $H$ induces in polynomial time a max-cost cut in $G$. $\Box$

We can now present our $\mathcal{PLS}$-completeness results. A graph has *degree* 3 if all vertices are connected to at most 3 other vertices each.

**Theorem 4.2** *The problem of finding stable states in Hopfield networks of degree 3 is $\mathcal{PLS}$-complete.*

PROOF: (Sketch.) By Lemma 4.1, it suffices to prove that the set of degree-3 graphs is pliable. Suppose $G = (V, E)$ is an arbitrary graph. Replace each degree-$k$ vertex $x \in V$ by a path consisting of $k$ vertices, and attach each edge incident with $v$ by a new edge incident with one of the vertices in the path. Figure 1 shows an example of this embedding. $\Box$

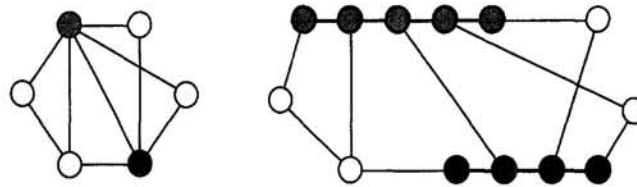

Figure 1: A guest graph of degree 5 (left), and the corresponding host of degree 3 (right). Shading indicates the high-degree nodes that were embedded into paths. All other nodes were embedded into single nodes.

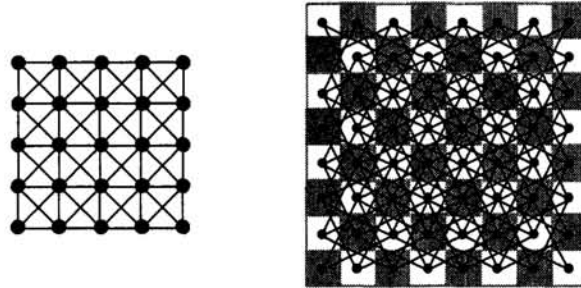

Figure 2: An 8-neighbor mesh with 25 vertices (left), and the 8 × 8 knight's graph superimposed on an 8 × 8 board (right).

The *8-neighbor mesh* is the degree-8 graph $G = (V, E)$ defined as follows: $V = \{1, 2, \ldots, m\} \times \{1, 2, \ldots, n\}$, and vertex $(u, v)$ is connected to vertices $(u, v \pm 1)$, $(u \pm 1, v)$, $(u \pm 1, v \pm 1)$. Figure 2 shows an 8-neighbor mesh with 25 vertices.

**Theorem 4.3** *The problem of finding stable states in Hopfield networks on the 8-neighbor mesh is $\mathcal{PLS}$-complete.*

PROOF: (Sketch.) By Lemma 4.1, it suffices to prove that the 8-neighbor mesh is pliable. An arbitrary graph can be embedded on an 8-neighbor mesh by mapping each node to a set of consecutive nodes in the bottom row of the grid, and mapping edges to disjoint rectilinear paths which use the diagonal edges of the grid for crossovers. □

The *knight's graph* for an $n \times n$ chessboard is the graph $G = (V, E)$ where $V = \{(i, j) \mid 1 \le i, j \le n\}$, and $E = \{((i, j), (k, \ell)) \mid \{|i - k|, |j - \ell|\} = \{1, 2\}\}$. That is, there is a vertex for every square of the board and an edge between two vertices exactly when there is a knight's move from one to the other. For example, Figure 2 shows the knight's graph for the 8 × 8 chessboard. Takefuji and Lee [15] (see also Parberry [12]) use the dual of the knight's graph for a Hopfield-style network to solve the knight's tour problem. That is, they have a vertex $v_e$ for each edge $e$ of the knight's graph, and an edge between two vertices $v_d$ and $v_e$ when $d$ and $e$ share a common vertex in the knight's graph.

**Theorem 4.4** *The problem of finding stable states in Hopfield networks on the dual of the knight's graph is $\mathcal{PLS}$-complete.*

PROOF: (Sketch.) By Lemma 4.1, it suffices to prove that the dual of the knight's graph is pliable. It can be shown that the knight's graph is pliable using the technique of Theorem 4.3. It can also be proved that if a set $S$ of graphs is pliable, then the set consisting of the duals of graphs in $S$ is also pliable. □

The *hypercube* is the graph with $2^d$ nodes for some $d$, labelled with the binary representations of the $d$-bit natural numbers, in which two nodes are connected by an edge iff their labels differ in exactly one bit. The hypercube is an important graph for parallel computation (see, for example, Leighton [8], and Parberry [10]).

**Theorem 4.5** *The problem of finding stable states in Hopfield networks on the hypercube is $\mathcal{PLS}$-complete.*

PROOF: (Sketch.) By Lemma 4.1, it suffices to prove that the hypercube is pliable. Since the "$\leq_e$" relation is transitive, it further suffices by Theorem 4.2 to show that the set of degree-3 graphs is polynomial-time embeddable into the hypercube. To embed a degree-3 graph $G$ into the hypercube, first break it into a degree-1 graph $G_1$ and a degree-2 graph $G_2$. Since $G_2$ consists of cycles, paths, and disconnected vertices, it can easily be embedded into a hypercube (since a hypercube is rich in cycles). $G_1$ can be viewed as a permutation of vertices in $G$ and can hence be realized using a hypercube implementation of Waksman's permutation network [16]. □

We conclude by stating $\mathcal{PLS}$-completeness results for three more graphs that are important in the parallel computing literature the *butterfly* (see, for example, Leighton [8]), the *cube-connected cycles* (Preparata and Vuillemin [13]), and the *shuffle-exchange* (Stone [14]). The proofs use Lemma 4.1 and Theorem 4.5, and are omitted for conciseness.

**Theorem 4.6** *The problem of finding stable states in Hopfield networks on the butterfly, the cube-connected cycles, and the shuffle-exchange is $\mathcal{PLS}$-complete.*

## Conclusion

Are Hopfield networks faster than conventional computers? The answer seems to be that it depends on the interconnection graph of the Hopfield network. Conventional nonimitative algorithms can be exponentially faster than planar Hopfield networks. The theory of $\mathcal{PLS}$-completeness shows us that such an exponential separation result is unlikely not only for nonplanar graphs, but even for simple nonplanar graphs such as bipartite graphs, graphs of degree 3, the dual of the knight's graph, the 8-neighbor mesh, the hypercube, the butterfly, the cube-connected cycles, and the shuffle-exchange graph.

## Acknowledgements

The research described in this paper was supported by the National Science Foundation under grant number CCR–9302917, and by the Air Force Office of Scientific

Research, Air Force Systems Command, USAF, under grant number F49620-93-1-0100.

## References

[1] J. Bruck and J. W. Goodman. A generalized convergence theorem for neural networks. *IEEE Transactions on Information Theory*, 34(5):1089–1092, 1988.

[2] E. Goles and S. Martinez. Exponential transient classes of symmetric neural networks for synchronous and sequential updating. *Complex Systems*, 3:589–597, 1989.

[3] F. Hadlock. Finding a maximum cut of a planar graph in polynomial time. *SIAM Journal on Computing*, 4(3):221–225, 1975.

[4] A. Haken and M. Luby. Steepest descent can take exponential time for symmetric conenction networks. *Complex Systems*, 2:191–196, 1988.

[5] J.-W. Hong, K. Mehlhorn, and A.L. Rosenberg. Cost tradeoffs in graph embeddings. *Journal of the ACM*, 30:709–728, 1983.

[6] J. J. Hopfield. Neural networks and physical systems with emergent collective computational abilities. *Proc. National Academy of Sciences*, 79:2554–2558, April 1982.

[7] D. S. Johnson, C. H. Papadimitriou, and M. Yannakakis. How easy is local search? In *26th Annual Symposium on Foundations of Computer Science*, pages 39–42. IEEE Computer Society Press, 1985.

[8] F. T. Leighton. *Introduction to Parallel Algorithms and Architectures: Arrays · Trees · Hypercubes*. Morgan Kaufmann, 1992.

[9] C. H. Papadimitriou, A. A. Schäffer, and M. Yannakakis. On the complexity of local search. In *Proceedings of the Twenty Second Annual ACM Symposium on Theory of Computing*, pages 439–445. ACM Press, 1990.

[10] I. Parberry. *Parallel Complexity Theory*. Research Notes in Theoretical Computer Science. Pitman Publishing, London, 1987.

[11] I. Parberry. *Circuit Complexity and Neural Networks*. MIT Press, 1994.

[12] I. Parberry. Scalability of a neural network for the knight's tour problem. *Neurocomputing*, 12:19–34, 1996.

[13] F. P. Preparata and J. Vuillemin. The cube-connected cycles: A versatile network for parallel computation. *Communications of the ACM*, 24(5):300–309, 1981.

[14] H. S. Stone. Parallel processing with the perfect shuffle. *IEEE Transactions on Computers*, C-20(2):153–161, 1971.

[15] Y. Takefuji and K. C. Lee. Neural network computing for knight's tour problems. *Neurocomputing*, 4(5):249–254, 1992.

[16] A. Waksman. A permutation network. *Journal of the ACM*, 15(1):159–163, January 1968.
